# A Short-Term Memory Architecture for the Learning of Morphophonemic Rules

**Michael Gasser and Chan-Do Lee**
Computer Science Department
Indiana University
Bloomington, IN 47405

## Abstract

Despite its successes, Rumelhart and McClelland's (1986) well-known approach to the learning of morphophonemic rules suffers from two deficiencies: (1) It performs the artificial task of associating forms with forms rather than perception or production. (2) It is not constrained in ways that humans learners are. This paper describes a model which addresses both objections. Using a simple recurrent architecture which takes both forms and "meanings" as inputs, the model learns to generate verbs in one or another "tense", given arbitrary meanings, and to recognize the tenses of verbs. Furthermore, it fails to learn reversal processes unknown in human language.

## 1 BACKGROUND

In the debate over the power of connectionist models to handle linguistic phenomena, considerable attention has been focused on the learning of simple morphological rules. It is a straightforward matter in a symbolic system to specify how the meanings of a stem and a bound morpheme combine to yield the meaning of a whole word and how the form of the bound morpheme depends on the shape of the stem. In a distributed connectionist system, however, where there may be no explicit morphemes, words, or rules, things are not so simple.

The most important work in this area has been that of Rumelhart and McClelland (1986), together with later extensions by Marchman and Plunkett (1989). The networks involved were trained to associate English verb stems with the corresponding past-tense forms, successfully generating both regular and irregular forms and generalizing to novel inputs. This work established that rule-like linguistic behavior

could be achieved in a system with no explicit rules. However, it did have important limitations, among them the following:

1. The representation of linguistic form was inadequate. This is clear, for example, from the fact that distinct lexical items may be associated with identical representations (Pinker & Prince, 1988).

2. The model was trained on an artificial task, quite unlike the perception and production that real hearers and speakers engage in. Of course, because it has no semantics, the model also says nothing about the issue of compositionality.

One consequence of both of these shortcomings is that there are few constraints on the kinds of processes that can be learned.

In this paper we describe a model which addresses these objections to the earlier work on morphophonemic rule acquisition. The model learns to generate forms in one or another "tense", given arbitrary patterns representing "meanings", and to yield the appropriate tense, given forms. The network sees linguistic forms one segment at a time, saving the context in a short-term memory. This style of representation, together with the more realistic tasks that the network is faced with, results in constraints on what can be learned. In particular, the system experiences difficulty learning reversal processes which do not occur in human language and which were easily accommodated by the earlier models.

## 2   SHORT-TERM MEMORY AND PREDICTION

Language takes place in time, and at some point, systems that learn and process language have to come to grips with this fact by accepting input in sequential form. Sequential models require some form of short-term memory (STM) because the decisions that are made depend on context. There are basically two options, window approaches, which make available stretches of input events all at once, and dynamic memory approaches (Port, 1990), which offer the possibility of a recoded version of past events. Networks with recurrent connections have the capacity for dynamic memory. We make use of a variant of a simple recurrent network (Elman, 1990), which is a pattern associator with recurrent connections on its hidden layer. Because the hidden layer receives input from itself as well as from the units representing the current event, it can function as a kind of STM for sequences of events.

Elman has shown how networks of this type can learn a great deal about the structure of the inputs when trained on the simple, unsupervised task of predicting the next input event. We are interested in what can be expected from such a network that is given a single phonological segment (hereafter referred to as a *phone*) at a time and trained to predict the next phone. If a system could learn to do this successfully, it would have a left-to-right version of what phonologists call *phonotactics*; that is, it would have knowledge of what phones tend to follow other phones in given contexts. Since word recognition and production apparently build on phonotactic knowledge of the language (Church, 1987), training on the prediction task might provide a way of integrating the two processes within a single network.

## 3   ARCHITECTURE

The type of network we work with is shown in Figure 1. Both its inputs and

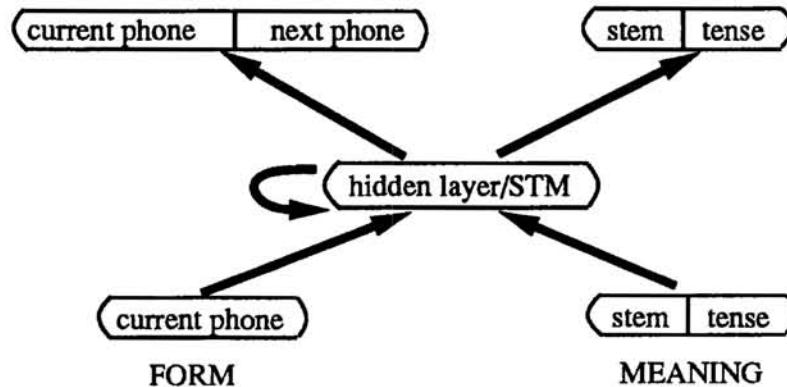

Figure 1: Network Architecture

outputs include FORM, that is, an individual phone, and what we'll call MEANING, that is, a pattern representing the stem of the word to be recognized or produced and a single unit representing a grammatical feature such as PAST or PRESENT. In fact, the meaning patterns have no real semantics, but like real meanings, they are arbitrarily assigned to the various morphemes and thus convey nothing about the phonological realization of the stem and grammatical feature. The network is trained both to auto-associate the current phone and predict the next phone.

The word recognition task corresponds to being given phone inputs (together with a default pattern on the meaning side) and generating meaning outputs. The meaning outputs are copied to the input meaning layer on each time step. While networks trained in this way can learn to recognize the words they are trained on, we have not been able to get them to generalize well. Networks which are expected only to output the grammatical feature, however, do generalize, as we shall see.

The word production task corresponds to being given a constant meaning input and generating form output. Following an initial default phone pattern, the phone input is what was predicted on the last time step. Again, however, though such a network does fine on the training set, it does not generalize well to novel inputs. We have had more success with a version using "teacher forcing". Here the correct current phone is provided on the input at each time step.

## 4   SIMULATIONS

### 4.1   STIMULI

We conducted a set of experiments to test the effectiveness of this architecture for the learning of morphophonemic rules. Input words were composed of sequences of phones in an artificial language. Each of the 15 possible phones was represented by a pattern over a set of 8 phonetic features. For each simulation, a set of 20 words was generated randomly from the set of possible words. Twelve of these were

designated "training" words, 8 "test" words.

For each of these basic words, there was an associated inflected form. For each simulation, one of a set of 9 rules was used to generate the inflected form: (1) suffix (+ assimilation) (gip→gips, gib→gibz), (2) prefix (+ assimilation) (gip→zgip, kip→skip), (3) gemination (iga→igga), (4) initial deletion (gip→ip), (5) medial deletion (ipka→ipa), (6) final deletion (gip→gi), (7) tone change (gip→gìp), (8) Pig Latin (gip→ipge), and (9) reversal (gip→pig).

In the two assimilation cases, the suffix or prefix agreed with the preceding or following phone on the voice feature. In the suffixing example, p is followed by s because it is voiceless, b by z because it is voiced. In the prefixing example, g is preceded by z because it is voiced, k by s because it is voiceless. Because the network is trained on prediction, these two rules are not symmetric. It would not be surprising if such a network could learn to generate a final phone which agrees in voicing with the phone preceding it. But in the prefixing case, the network must choose the correct prefix before it has seen the phone with which it is to agree in voicing. We thought this would still be possible, however, because the network also receives meaning input representing the stem of the word to be produced.

We hoped that the network would succeed on rule types which are common in natural languages and fail on those which are rare or non-existent. Types 1–4 are relatively common, types 5–7 infrequent or rare, type 8 apparently known only in language games, and type 9 apparently non-occurring.

For convenience, we will refer to the uninflected form of a word as the "present" and the inflected form as the "past tense" of the word in question. Each input word consisted of a present or past tense form preceded and followed by a word boundary pattern composed of zeroes. Meaning patterns consisted of an arbitrary pattern across a set of 6 "stem" units, representing the meaning of the "stem" of one of the 20 input words, plus a single bit representing the "tense" of the input word, that is, present or past.

## 4.2   TRAINING

During training each of the training words was presented in both present and past forms, while the test words appeared in the present form only. Each of the 32 separate words was trained in both the recognition and production directions.

For recognition training, the words were presented, one phone at a time, on the form input units. The appropriate pattern was also provided on the stem meaning units. Targets specified the current phone, next phone, and complete meaning. Thus the network was actually being trained to generate only the tense portion of the meaning for each word. The activation on the tense output unit was copied to the tense input unit following each time step.

For production training, the stem and grammatical feature were presented on the lexical input layer and held constant throughout the word. The phones making up the word were presented one at a time beginning with the initial word boundary, and the network was expected to predict the next phone in each case.

There were 10 separate simulations for each of the 9 inflectional rules. Pilot runs

Table 1: Results of Recognition and Production Tests

| | RECOGNITION | PRODUCTION | |
| | % tenses correct | % segments correct | % affixes correct |
|---|---|---|---|
| Suffix | 79 | 82 | 83 |
| Prefix | 76 | 62 | 76 |
| Tone change | 99 | 98 | – |
| Gemination | 90 | 74 | 42 |
| Deletion | 67 | 31 | – |
| Pig Latin | 61 | 27 | – |
| Reversal | 13 | 23 | – |

were used to find estimates of the best hidden layer size. This varied between 16 and 26. Training continued until the mean sum-of-squares error was less than 0.05. This normally required between 50 and 100 epochs. Then the connection weights were frozen, and the network was tested in both the recognition and production directions on the past tense forms of the test words.

## 4.3  RESULTS

In all cases, the network learned the training set quite successfully (at least 95% of the phones for production and 96% of the tenses for recognition). Results for the recognition and production of past-tense forms of test words are shown in Table 1. For recognition, chance is 37.5%. For production, the network's output on a given time step was considered to be that phone which was closest to the pattern on the phone output units.

## 5  DISCUSSION

### 5.1  AFFIXATION AND ASSIMILATION

The model shows clear evidence of having learned morphophonemic rules which it uses in both the production and perception directions. And the degree of mastery of the rules, at least for production, mirrors the extent to which the types of rules occur in natural languages. Significantly, the net is able to generate appropriate forms even in the prefix case when a "right-to-left" (anticipatory) rule is involved. That is, the fact that the network is trained only on prediction does not limit it to left-to-right (perseverative) rules because it has access to a "meaning" which permits the required "lookahead" to the relevant feature on the phone following the prefix. What makes this interesting is the fact that the meaning patterns bear no relation to the phonology of the stems. The connections between the stem meaning input units and the hidden layer are being trained to encode the voicing feature even when, in the case of the test words, this was never required during training.

In any case, it is clear that right-to-left assimilation in a network such as this is more difficult to acquire than left-to-right assimilation, all else being equal. We are

unaware of any evidence that would support this, though the fact that prefixes are less common than suffixes in the world's languages (Hawkins & Cutler, 1988) means that there are at least fewer opportunities for the right-to-left process.

## 5.2  REVERSAL

What is it that makes the reversal rule, apparently difficult for human language learners, so difficult for the network? Consider what the network does when it is faced with the past-tense form of a verb trained only in the present. If the novel item took the form of a set rather than a sequence, it would be identical to the familiar present-tense form. What the network sees, however, is a sequence of phones, and its task is to predict the next. There is thus no sharing at all between the present and past forms and no basis for generalizing from the present to the past. Presented with the novel past form, it is more likely to base its response on similarity with a word containing a similar sequence of phones (e.g., gip and gif) than it is with the correct mirror-image sequence.

It is important to note, however, that difficulty with the reversal process does not necessarily presuppose the type of representations that result from training a simple recurrent net on prediction. Rather this depends more on the fact that the network is trained to map meaning to form and form to meaning, rather than form to form, as in the case of the Rumelhart and McClelland (1986) model. Any network of the former type which represents linguistic form in such a way that the contexts of the phones are preserved is likely to exhibit this behavior.[1]

## 6  LIMITATIONS AND EXTENSIONS

Despite its successes, this model is far from an adequate account of the recognition and production of words in natural language. First, although networks of the type studied here are capable of yielding complete meanings given words and complete words given meanings, they have difficulty when expected to respond to novel forms or combinations of known meanings. In the simulations, we asked the network to recognize only the grammatical morpheme in a novel word, and in production we kept it on track by giving it the correct input phone on each time step. It will be important to discover ways to make the system robust enough to respond appropriately to novel forms and combinations of meanings.

Equally important is the ability of the model to handle more complex phonological processes. Recently Lakoff (1988) and Touretzky and Wheeler (1990) have developed connectionist models to deal with complicated interacting phonological rules. While these models demonstrate that connectionism offers distinct advantages to conventional serial approaches to phonology, they do not learn phonology (at least not in a connectionist way), and they do not yet accommodate perception.

We believe that the performance of the model will be significantly improved by the capacity to make reference directly to units larger than the phone. We are currently investigating an architecture consisting of a hierarchy of networks of the type described here, each trained on the prediction task at a different time scale.

# 7   CONCLUSIONS

It is by now clear that a connectionist system can be trained to exhibit rule-like behavior. What is not so clear is whether networks can discover how to map elements of form onto elements of meaning and to use this knowledge to interpret and generate novel forms. It has been argued (Fodor & Pylyshyn, 1988) that this behavior requires the kind of constituency which is not available to networks making use of distributed representations.

The present study is one attempt to demonstrate that networks are not limited in this way. We have shown that, given "meanings" and temporally distributed representations of words, a network can learn to isolate stems and the realizations of grammatical features, associate them with their meanings, and, in a somewhat limited sense, use this knowledge to produce and recognize novel forms. In addition, the nature of the training task constrains the system in such a way that rules which are rare or non-occurring in natural language are not learned.

**References**

Church, K. W. (1987). Phonological parsing and lexical retrieval. *Cognition, 25,* 53–69.

Elman, J. (1990). Finding structure in time. *Cognitive Science, 14,* 179–211.

Fodor, J., & Pylyshyn, Z. (1988). Connectionism and cognitive architecture: A critical analysis. *Cognition, 28,* 3–71.

Hawkins, J. A., & Cutler, A. (1988). Psychological factors in morphological asymmetry. In J. A. Hawkins (Ed.), *Explaining language universals* (pp. 280-317). Oxford: Basil Blackwell.

Lakoff, G. (1988). *Cognitive phonology.* Paper presented at the Annual Meeting of the Linguistics Society of America.

Marchman, V., & Plunkett, K. (1989). Token frequency and phonological predictability in a pattern association network: Implications for child language acquisition. *Proceedings of the Annual Conference of the Cognitive Science Society, 11,* 179–187.

Pinker, S., & Prince, A. (1988). On language and connectionism: Analysis of a parallel distributed processing model of language acquisition. *Cognition, 28,* 73–193.

Port, R. (1990). Representation and recognition of temporal patterns. *Connection Science, 2,* 151–176.

Rumelhart, D., & McClelland, J. (1986). On learning the past tense of English verbs. In J. L. McClelland & D. E. Rumelhart (Eds.), *Parallel Distributed Processing*, Vol. 2 (pp. 216–271). Cambridge, MA: MIT Press.

Touretzky, D. and Wheeler, D. (1990). A computational basis for phonology. In D. S. Touretzky (Ed.), *Advances in Neural Information Processing Systems 2*, San Mateo, CA: Morgan Kaufmann.

## Footnotes

[1]We are indebted to Dave Touretzky for helping to clarify this issue.
